# A Bayesian Framework for Tilt Perception and Confidence

**Odelia Schwartz**
HHMI and Salk Institute
La Jolla, CA 92014
odelia@salk.edu

**Terrence J. Sejnowski**
HHMI and Salk Institute
La Jolla, CA 92014
terry@salk.edu

**Peter Dayan**
Gatsby, UCL
17 Queen Square, London
dayan@gatsby.ucl.ac.uk

## Abstract

The misjudgement of tilt in images lies at the heart of entertaining visual illusions and rigorous perceptual psychophysics. A wealth of findings has attracted many mechanistic models, but few clear computational principles. We adopt a Bayesian approach to perceptual tilt estimation, showing how a smoothness prior offers a powerful way of addressing much confusing data. In particular, we faithfully model recent results showing that *confidence* in estimation can be systematically affected by the same aspects of images that affect bias. Confidence is central to Bayesian modeling approaches, and is applicable in many other perceptual domains.

Perceptual anomalies and illusions, such as the misjudgements of motion and tilt evident in so many psychophysical experiments, have intrigued researchers for decades.[1–3] A Bayesian view[4–8] has been particularly influential in models of motion processing, treating such anomalies as the normative product of prior information (often statistically codifying Gestalt laws) with likelihood information from the actual scenes presented. Here, we expand the range of statistically normative accounts to tilt estimation, for which there are classes of results (on estimation confidence) that are so far not available for motion.

The tilt illusion arises when the perceived tilt of a center target is misjudged (*ie bias*) in the presence of flankers. Another phenomenon, called Crowding, refers to a loss in the confidence (*ie sensitivity*) of perceived target tilt in the presence of flankers. Attempts have been made to formalize these phenomena quantitatively. Crowding has been modeled as compulsory feature pooling (*ie* averaging of orientations), ignoring spatial positions.[9, 10] The tilt illusion has been explained by lateral interactions[11, 12] in populations of orientation-tuned units; and by *calibration*.[13]

However, most models of this form cannot explain a number of crucial aspects of the data. First, the *geometry* of the positional arrangement of the stimuli affects attraction versus repulsion in bias, as emphasized by Kapadia *et al*[14] (figure 1A), and others.[15, 16] Second, Solomon et al. recently measured bias *and* sensitivity simultaneously.[11] The rich and surprising range of sensitivities, far from flat as a function of flanker angles (figure 1B), are outside the reach of standard models. Moreover, current explanations do not offer a computational account of tilt perception as the outcome of a normative inference process.

Here, we demonstrate that a Bayesian framework for orientation estimation, with a prior favoring smoothness, can naturally explain a range of seemingly puzzling tilt data. We explicitly consider both the geometry of the stimuli, and the issue of confidence in the esti-

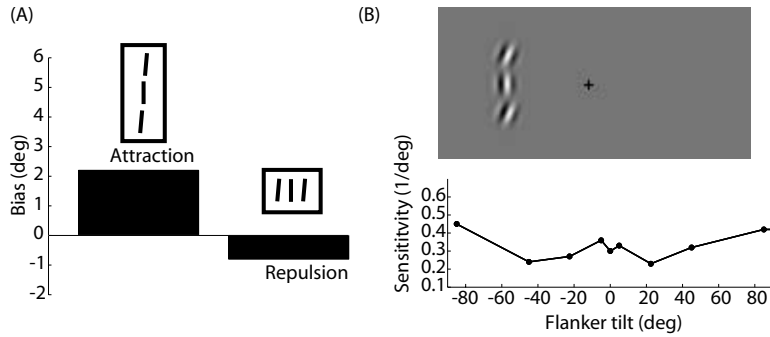

**Figure 1:** Tilt biases and sensitivities in visual perception. **(A)** Kapadia *et al* demonstrated the importance of geometry on tilt bias, with bar stimuli in the fovea (and similar results in the periphery). When 5 degrees clockwise flankers are arranged colinearly, the center target appears attracted in the direction of the flankers; when flankers are lateral, the target appears repulsed. Data are an average of 5 subjects.[14] **(B)** Solomon *et al* measured both biases and sensitivities for gratings in the visual periphery.[11] On the top are example stimuli, with flankers tilted 22.5 degrees clockwise. This constitutes the classic tilt illusion, with a repulsive bias percept. In addition, sensitivities vary as a function of flanker angles, in a systematic way (even in cases when there are no biases at all). Sensitivities are given in units of the inverse of standard deviation of the tilt estimate. More detailed data for both experiments are shown in the results section.

mation. Bayesian analyses have most frequently been applied to bias. Much less attention has been paid to the equally important phenomenon of sensitivity. This aspect of our model should be applicable to other perceptual domains.

In section 1 we formulate the Bayesian model. The prior is determined by the principle of creating a smooth contour between the target and flankers. We describe how to extract the bias and sensitivity. In section 2 we show experimental data of Kapadia *et al* and Solomon *et al*, alongside the model simulations, and demonstrate that the model can account for both geometry, and bias and sensitivity measurements in the data. Our results suggest a more unified, rational, approach to understanding tilt perception.

## 1   Bayesian model

Under our Bayesian model, inference is controlled by the posterior distribution over the tilt of the target element. This comes from the combination of a prior favoring smooth configurations of the flankers and target, and the likelihood associated with the actual scene. A complete distribution would consider all possible angles and relative spatial positions of the bars, and marginalize the posterior over all but the tilt of the central element. For simplicity, we make two benign approximations: conditionalizing over (*ie* clamping) the *angles* of the flankers, and exploring only a small neighborhood of their positions. We now describe the steps of inference.

**Smoothness prior:** Under these approximations, we consider a given *actual* configuration (see fig 2A) of flankers $f_1 = (\phi_1, x_1)$, $f_2 = (\phi_2, x_2)$ and center target $c = (\phi_c, x_c)$, arranged from top to bottom. We have to generate a prior over $\phi_c$ and $\delta_1 = x_1 - x_c$ and $\delta_2 = x_2 - x_c$ based on the principle of smoothness. As a less benign approximation, we do this in two stages: articulating a principle that determines a single optimal configuration; and generating a prior as a mixture of a Gaussian about this optimum and a uniform distribution, with the mixing proportion of the latter being determined by the smoothness of the optimum.

Smoothness has been extensively studied in the computer vision literature.[17–20] One widely

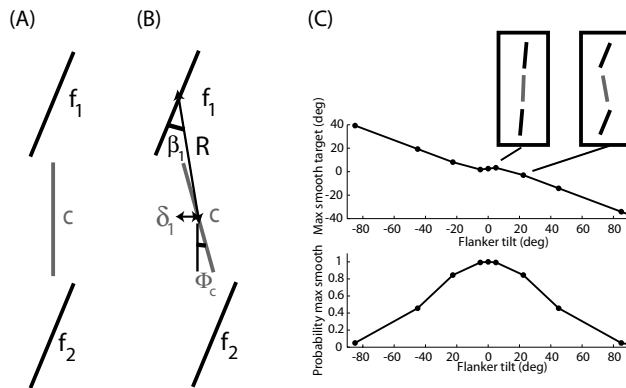

**Figure 2:** Geometry and smoothness for flankers, $f_1$ and $f_2$, and center target, $c$. **(A)** Example *actual* configuration of flankers and target, aligned along the $y$ axis from top to bottom. **(B)** The elastica procedure can rotate the target angle (to $\Phi_c$) and shift the relative flanker and target positions on the $x$ axis (to $\delta_1$ and $\delta_2$) in its search for the maximally smooth solution. Small spatial shifts (up to $1/15$ the size of $R$) of positions are allowed, but positional shift is over-emphasized in the figure for visibility. **(C)** Top: center tilt that results in maximal smoothness, as a function of flanker tilt. Boxed cartoons show examples for given flanker tilts, of the optimally smooth configuration. Note attraction of target towards flankers for small flanker angles; here flankers and target are positioned in a nearly colinear arrangement. Note also repulsion of target away from flankers for intermediate flanker angles. Bottom: $P[c, f_1, f_2]$ for center tilt that yields maximal smoothness. The $y$ axis is normalized between 0 and 1.

used principle, *elastica*, known even to Euler, has been applied to contour completion[21] and other computer vision applications.[17] The basic idea is to find the curve with minimum energy (*ie*, square of curvature). Sharon *et al*[19] showed that the elastica function can be well approximated by a number of simpler forms. We adopt a version that Leung and Malik[18] adopted from Sharon *et al*.[19] We assume that the probability for completing a smooth curve, can be factorized into two terms:

$$P[c, f_1, f_2] = G(c, f_1)G(c, f_2) \tag{1}$$

with the term $G(c, f_1)$ (and similarly, $G(c, f_2)$) written as:

$$G(c, f_1) = \exp(-\frac{R}{\sigma_R} - \frac{D_\beta}{\sigma_\beta}) \qquad \text{where} \qquad D_\beta = \beta_1^2 + \beta_c^2 - \beta_1\beta_c \tag{2}$$

and $\beta_1$ (and similarly, $\beta_c$) is the angle between the orientation at $f_1$, and the line joining $f_1$ and $c$. The distance between the centers of $f_1$ and $c$ is given by $R$. The two constants, $\sigma_\beta$ and $\sigma_R$, control the relative contribution to smoothness of the angle versus the spatial distance. Here, we set $\sigma_\beta = 1$, and $\sigma_R = 1.5$. Figure 2B illustrates an example geometry, in which $\phi_c$, $\delta_1$, and $\delta_2$, have been shifted from the actual scene (of figure 2A).

We now estimate the smoothest solution for given configurations. Figure 2C shows for given flanker tilts, the center tilt that yields maximal smoothness, and the corresponding probability of smoothness. For near vertical flankers, the spatial lability leads to very weak attraction and high probability of smoothness. As the flanker angle deviates farther from vertical, there is a large repulsion, but also lower probability of smoothness. These observations are key to our model: the maximally smooth center tilt will influence attractive and repulsive interactions of tilt estimation; the probability of smoothness will influence the relative weighting of the prior versus the likelihood.

From the smoothness principle, we construct a two dimensional prior (figure 3A). One dimension represents tilt, the other dimension, the overall positional shift between target

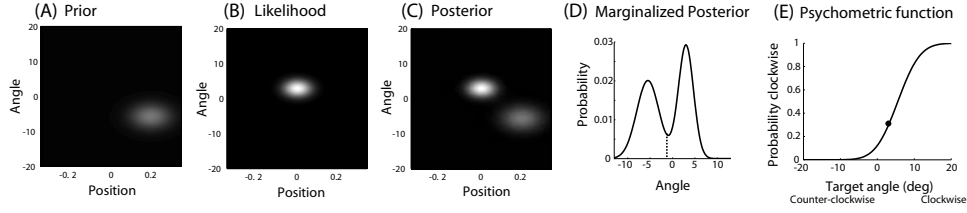

**Figure 3:** Bayes model for example flankers and target. **(A)** Prior 2D distribution for flankers set at 22.5 degrees (note repulsive preference for -5.5 degrees). **(B)** Likelihood 2D distribution for a target tilt of 3 degrees; **(C)** Posterior 2D distribution. All 2D distributions are drawn on the same grayscale range, and the presence of a larger baseline in the prior causes it to appear more dimmed. **(D)** Marginalized posterior, resulting in 1D distribution over tilt. Dashed line represents the mean, with slight preference for negative angle. **(E)** For this target tilt, we calculate probability clockwise, and obtain one point on psychometric curve.

and flankers (called 'position'). The prior is a 2D Gaussian distribution, sat upon a constant baseline.[22] The Gaussian is centered at the estimated smoothest target angle and relative position, and the baseline is determined by the probability of smoothness. The baseline, and its dependence on the flanker orientation, is a key difference from Weiss *et al*'s Gaussian prior for smooth, slow motion. It can be seen as a mechanism to allow segmentation (see Posterior description below). The standard deviation of the Gaussian is a free parameter.

**Likelihood:** The likelihood over tilt and position (figure 3B) is determined by a 2D Gaussian distribution with an added baseline.[22] The Gaussian is centered at the actual target tilt; and at a position taken as zero, since this is the actual position, to which the prior is compared. The standard deviation and baseline constant are free parameters.

**Posterior and marginalization:** The posterior comes from multiplying likelihood and prior (figure 3C) and then marginalizing over position to obtain a 1D distribution over tilt. Figure 3D shows an example in which this distribution is bimodal. Other likelihoods, with closer agreement between target and smooth prior, give unimodal distributions. Note that the bimodality is a direct consequence of having an added baseline to the prior and likelihood (if these were Gaussian without a baseline, the posterior would always be Gaussian). The viewer is effectively assessing whether the target is associated with the same object as the flankers, and this is reflected in the baseline, and consequently, in the bimodality, and confidence estimate. We define $\alpha$ as the mean angle of the 1D posterior distribution (*eg*, value of dashed line on the $x$ axis), and $\beta$ as the height of the probability distribution at that mean angle (*eg*, height of dashed line). The term $\beta$ is an indication of confidence in the angle estimate, where for larger values we are more certain of the estimate.

**Decision of probability clockwise:** The probability of a clockwise tilt is estimated from the marginalized posterior:

$$P = \frac{1}{1 + \exp\left(\frac{-\alpha .* k}{-\log(\beta + \eta)}\right)} \tag{3}$$

where $\alpha$ and $\beta$ are defined as above, $k$ is a free parameter and $\eta$ a small constant. Free parameters are set to a single constant value for all flanker and center configurations. Weiss *et al* use a similar compressive nonlinearity, but without the term $\beta$. We also tried a decision function that integrates the posterior, but the resulting curves were far from the sigmoidal nature of the data.

**Bias and sensitivity:** For one target tilt, we generate a single probability and therefore a single point on the psychometric function relating tilt to the probability of choosing clockwise. We generate the full psychometric curve from all target tilts and fit to it a cumulative

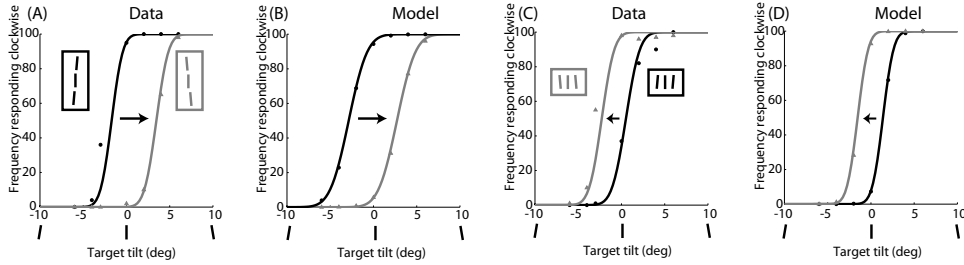

**Figure 4:** Kapadia *et al* data,[14] versus Bayesian model. Solid lines are fits to a cumulative Gaussian distribution. **(A)** Flankers are tilted 5 degrees clockwise (black curve) or anti-clockwise (gray) of vertical, and positioned spatially in a colinear arrangement. The center bar appears tilted in the direction of the flankers (attraction), as can be seen by the attractive shift of the psychometric curve. The boxed stimuli cartoon illustrates a vertical target amidst the flankers. **(B)** Model for colinear bars also produces attraction. **(C)** Data and **(D)** model for lateral flankers results in repulsion. All data are collected in the fovea for bars.

Gaussian distribution $N(\mu, \sigma)$ (figure 3E). The mean $\mu$ of the fit corresponds to the bias, and $\frac{1}{\sigma}$ to the sensitivity, or confidence in the bias. The fit to a cumulative Gaussian and extraction of these parameters exactly mimic psychophysical procedures.[11]

## 2 Results: data versus model

We first consider the geometry of the center and flanker configurations, modeling the full psychometric curve for colinear and parallel flanks (recall that figure 1A showed summary biases). Figure 4A;B demonstrates attraction in the data and model; that is, the psychometric curve is shifted towards the flanker, because of the nature of smooth completions for colinear flankers. Figure 4C;D shows repulsion in the data and model. In this case, the flankers are arranged laterally instead of colinearly. The smoothest solution in the model arises by shifting the target estimate away from the flankers. This shift is rather minor, because the configuration has a low probability of smoothness (similar to figure 2C), and thus the prior exerts only a weak effect.

The above results show examples of changes in the psychometric curve, but do not address both bias and, particularly, sensitivity, across a whole range of flanker configurations. Figure 5 depicts biases and sensitivity from Solomon *et al*, versus the Bayes model. The data are shown for a representative subject, but the qualitative behavior is consistent across all subjects tested. In figure 5A, bias is shown, for the condition that both flankers are tilted at the same angle. The data exhibit small attraction at near vertical flanker angles (this arrangement is close to colinear); large repulsion at intermediate flanker angles of 22.5 and 45 degrees from vertical; and minimal repulsion at large angles from vertical. This behavior is also exhibited in the Bayes model (Figure 5B). For intermediate flanker angles, the smoothest solution in the model is repulsive, and the effect of the prior is strong enough to induce a significant repulsion. For large angles, the prior exerts almost no effect.

Interestingly, sensitivity is far from flat in both data and model. In the data (Figure 5C), there is most loss in sensitivity at intermediate flanker angles of 22.5 and 45 degrees (*ie*, the subject is less certain); and sensitivity is higher for near vertical or near horizontal flankers. The model shows the same qualitative behavior (Figure 5D). In the model, there are two factors driving sensitivity: one is the probability of completing a smooth curvature for a given flanker configuration, as in Figure 2B; this determines the strength of the prior. The other factor is certainty in a particular center estimation; this is determined by $\beta$, derived from the posterior distribution, and incorporated into the decision stage of the model

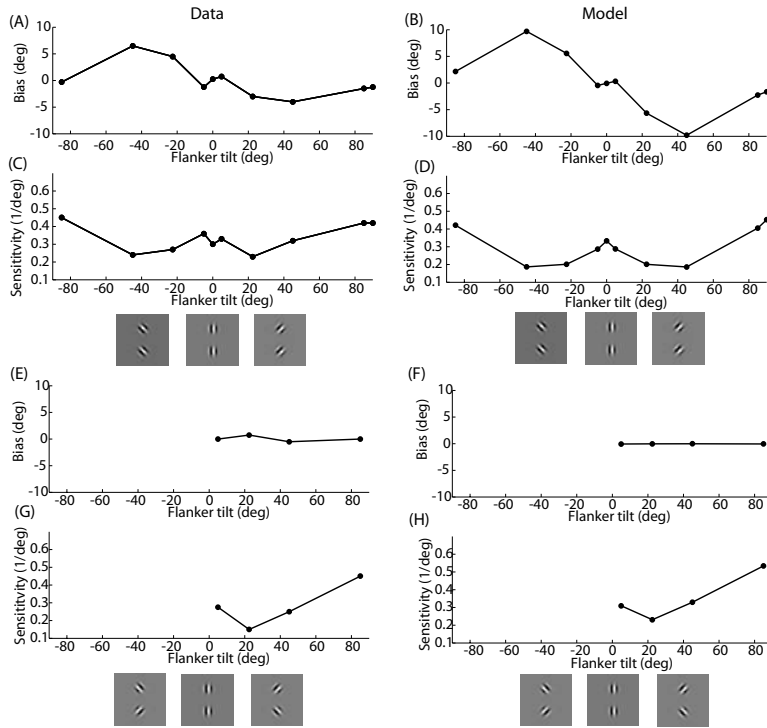

**Figure 5:** Solomon *et al* data[11] (subject FF), versus Bayesian model. **(A)** Data and **(B)** model biases with same-tilted flankers; **(C)** Data and **(D)** model sensitivities with same-tilted flankers; **(E;G)** data and **(F;H)** model as above, but for opposite-tilted flankers (note that opposite-tilted data was collected for less flanker angles). Each point in the figure is derived by fitting a cumulative Gaussian distribution $N(\mu, \sigma)$ to corresponding psychometric curve, and setting bias equal to $\mu$ and sensitivity to $\frac{1}{\sigma}$. In all experiments, flanker and target gratings are presented in the visual periphery. Both data and model stimuli are averages of two configurations, on the left hand side (9 O'clock position) and right hand side (3 O'clock position). The configurations are similar to Figure 1 (B), but slightly shifted according to an iso-eccentric circle, so that all stimuli are similarly visible in the periphery.

(equation 3). For flankers that are far from vertical, the prior has minimal effect because one cannot find a smooth solution (*eg*, the likelihood dominates), and thus sensitivity is higher. The low sensitivity at intermediate angles arises because the prior has considerable effect; and there is conflict between the prior (tilt, position), and likelihood (tilt, position). This leads to uncertainty in the target angle estimation . For flankers near vertical, the prior exerts a strong effect; but there is less conflict between the likelihood and prior estimates (tilt, position) for a vertical target. This leads to more confidence in the posterior estimate, and therefore, higher sensitivity. The only aspect that our model does not reproduce is the (more subtle) sensitivity difference between 0 and +/- 5 degree flankers.

Figure 5E-H depict data and model for opposite tilted flankers. The bias is now close to zero in the data (Figure 5E) and model (Figure 5F), as would be expected (since the maximally smooth angle is now always roughly vertical). Perhaps more surprisingly, the sensitivities continue to to be non-flat in the data (Figure 5G) and model (Figure 5H). This behavior arises in the model due to the strength of prior, and positional uncertainty. As before, there is most loss in sensitivity at intermediate angles.

Note that to fit Kapadia *et al*, simulations used a constant parameter of $k = 9$ in equation

3, whereas for the Solomon et al. simulations, $k = 2.5$. This indicates that, in our model, there was higher confidence in the foveal experiments than in the peripheral ones.

## 3   Discussion

We applied a Bayesian framework to the widely studied tilt illusion, and demonstrated the model on examples from two different data sets involving foveal and peripheral estimation. Our results support the appealing hypothesis that perceptual misjudgements are not a consequence of poor system design, but rather can be described as optimal inference.[4–8] Our model accounts correctly for both attraction and repulsion, determined by the smoothness prior and the geometry of the scene.

We emphasized the issue of estimation confidence. The dataset showing how confidence is affected by the same issues that affect bias,[11] was exactly appropriate for a Bayesian formulation; other models in the literature typically do not incorporate confidence in a thoroughly probabilistic manner. In fact, our model fits the confidence (and bias) data more proficiently than an account based on lateral interactions among a population of orientation-tuned cells.[11] Other Bayesian work, by Stocker *et al*,[6] utilized the full slope of the psychometric curve in fitting a prior and likelihood to motion data, but did not examine the issue of confidence. Estimation confidence plays a central role in Bayesian formulations as a whole. Understanding how priors affect confidence should have direct bearing on many other Bayesian calculations such as multimodal integration.[23]

Our model is obviously over-simplified in a number of ways. First, we described it in terms of tilts and spatial positions; a more complete version should work in the pixel/filtering domain.[18,19] We have also only considered two flanking elements; the model is extendible to a full-field surround, whereby smoothness operates along a range of geometric directions, and some directions are more (smoothly) dominant than others. Second, the prior is constructed by summarizing the maximal smoothness information; a more probabilistically correct version should capture the full probability of smoothness in its prior. Third, our model does not incorporate a formal noise representation; however, sensitivities could be influenced both by stimulus-driven noise and confidence. Fourth, our model does not address attraction in the so-called indirect tilt illusion, thought to be mediated by a different mechanism. Finally, we have yet to account for neurophysiological data within this framework, and incorporate constraints at the neural implementation level. However, versions of our computations are oft suggested for intra-areal and feedback cortical circuits; and smoothness principles form a key part of the association field connection scheme in Li's[24] dynamical model of contour integration in V1.

Our model is connected to a wealth of literature in computer vision and perception. Notably, occlusion and contour completion might be seen as the extreme example in which there is no likelihood information at all for the center target; a host of papers have shown that under these circumstances, smoothness principles such as *elastica* and variants explain many aspects of perception. The model is also associated with many studies on contour integration motivated by Gestalt principles;[25,26] and exploration of natural scene statistics and Gestalt,[27,28] including the relation to contour grouping within a Bayesian framework.[29,30] Indeed, our model could be modified to include a prior from natural scenes.

There are various directions for the experimental test and refinement of our model. Most pressing is to determine bias and sensitivity for different center and flanker contrasts. As in the case of motion, our model predicts that when there is more uncertainty in the center element, prior information is more dominant. Another interesting test would be to design a task such that the center element is actually part of a different figure and unrelated to the flankers; our framework predicts that there would be minimal bias, because of segmentation. Our model should also be applied to other tilt-based illusions such as the Fraser spiral

and Zöllner. Finally, our model can be applied to other perceptual domains;[31] and given the apparent similarities between the tilt illusion and the tilt after-effect, we plan to extend the model to adaptation, by considering smoothness in time as well as space.

**Acknowledgements** This work was funded by the HHMI (OS, TJS) and the Gatsby Charitable Foundation (PD). We are very grateful to Serge Belongie, Leanne Chukoskie, Philip Meier and Joshua Solomon for helpful discussions.

## References

[1] J J Gibson. Adaptation, after-effect, and contrast in the perception of tilted lines. *Journal of Experimental Psychology*, 20:553–569, 1937.

[2] C Blakemore, R H S Carpentar, and M A Georgeson. Lateral inhibition between orientation detectors in the human visual system. *Nature*, 228:37–39, 1970.

[3] J A Stuart and H M Burian. A study of separation difficulty: Its relationship to visual acuity in normal and amblyopic eyes. *American Journal of Ophthalmology*, 53:471–477, 1962.

[4] A Yuille and H H Bulthoff. Perception as bayesian inference. In Knill and Whitman, editors, *Bayesian decision theory and psychophysics*, pages 123–161. Cambridge University Press, 1996.

[5] Y Weiss, E P Simoncelli, and E H Adelson. Motion illusions as optimal percepts. *Nature Neuroscience*, 5:598–604, 2002.

[6] A Stocker and E P Simoncelli. Constraining a bayesian model of human visual speed perception. *Adv in Neural Info Processing Systems*, 17, 2004.

[7] D Kersten, P Mamassian, and A Yuille. Object perception as bayesian inference. *Annual Review of Psychology*, 55:271–304, 2004.

[8] K Kording and D Wolpert. Bayesian integration in sensorimotor learning. *Nature*, 427:244–247, 2004.

[9] L Parkes, J Lund, A Angelucci, J Solomon, and M Morgan. Compulsory averaging of crowded orientation signals in human vision. *Nature Neuroscience*, 4:739–744, 2001.

[10] D G Pelli, M Palomares, and N J Majaj. Crowding is unlike ordinary masking: Distinguishing feature integration from detection. *Journal of Vision*, 4:1136–1169, 2002.

[11] J Solomon, F M Felisberti, and M Morgan. Crowding and the tilt illusion: Toward a unified account. *Journal of Vision*, 4:500–508, 2004.

[12] J A Bednar and R Miikkulainen. Tilt aftereffects in a self-organizing model of the primary visual cortex. *Neural Computation*, 12:1721–1740, 2000.

[13] C W Clifford, P Wenderoth, and B Spehar. A functional angle on some after-effects in cortical vision. *Proc Biol Sci*, 1454:1705–1710, 2000.

[14] M K Kapadia, G Westheimer, and C D Gilbert. Spatial distribution of contextual interactions in primary visual cortex and in visual perception. *J Neurophysiology*, 4:2048–262, 2000.

[15] C C Chen and C W Tyler. Lateral modulation of contrast discrimination: Flanker orientation effects. *Journal of Vision*, 2:520–530, 2002.

[16] I Mareschal, M P Sceniak, and R M Shapley. Contextual influences on orientation discrimination: binding local and global cues. *Vision Research*, 41:1915–1930, 2001.

[17] D Mumford. Elastica and computer vision. In Chandrajit Bajaj, editor, *Algebraic geometry and its applications*. Springer Verlag, 1994.

[18] T K Leung and J Malik. Contour continuity in region based image segmentation. In *Proc. ECCV*, pages 544–559, 1998.

[19] E Sharon, A Brandt, and R Basri. Completion energies and scale. *IEEE Pat. Anal. Mach. Intell.*, 22(10), 1997.

[20] S W Zucker, C David, A Dobbins, and L Iverson. The organization of curve detection: coarse tangent fields. *Computer Graphics and Image Processing*, 9(3):213–234, 1988.

[21] S Ullman. Filling in the gaps: the shape of subjective contours and a model for their generation. *Biological Cybernetics*, 25:1–6, 1976.

[22] G E Hinton and A D Brown. Spiking boltzmann machines. *Adv in Neural Info Processing Systems*, 12, 1998.

[23] R A Jacobs. What determines visual cue reliability? *Trends in Cognitive Sciences*, 6:345–350, 2002.

[24] Z Li. A saliency map in primary visual cortex. *Trends in Cognitive Science*, 6:9–16, 2002.

[25] D J Field, A Hayes, and R F Hess. Contour integration by the human visual system: evidence for a local "association field". *Vision Research*, 33:173–193, 1993.

[26] J Beck, A Rosenfeld, and R Ivry. Line segregation. *Spatial Vision*, 4:75–101, 1989.

[27] M Sigman, G A Cecchi, C D Gilbert, and M O Magnasco. On a common circle: Natural scenes and gestalt rules. *PNAS*, 98(4):1935–1940, 2001.

[28] S Mahumad, L R Williams, K K Thornber, and K Xu. Segmentation of multiple salient closed contours from real images. *IEEE Pat. Anal. Mach. Intell.*, 25(4):433–444, 1997.

[29] W S Geisler, J S Perry, B J Super, and D P Gallogly. Edge co-occurence in natural images predicts contour grouping performance. *Vision Research*, 6:711–724, 2001.

[30] J H Elder and R M Goldberg. Ecological statistics of gestalt laws for the perceptual organization of contours. *Journal of Vision*, 4:324–353, 2002.

[31] S R Lehky and T J Sejnowski. Neural model of stereoacuity and depth interpolation based on a distributed representation of stereo disparity. *Journal of Neuroscience*, 10:2281–2299, 1990.
